# Modeling Nonlinear Dependencies in Natural Images using Mixture of Laplacian Distribution

**Hyun Jin Park and Te Won Lee**
Institute for Neural Computation, UCSD
9500 Gilman Drive, La Jolla, CA 92093-0523
{hjinpark, tewon}@ucsd.edu

## Abstract

Capturing dependencies in images in an unsupervised manner is important for many image processing applications. We propose a new method for capturing nonlinear dependencies in images of natural scenes. This method is an extension of the linear Independent Component Analysis (ICA) method by building a hierarchical model based on ICA and mixture of Laplacian distribution. The model parameters are learned via an EM algorithm and it can accurately capture variance correlation and other high order structures in a simple manner. We visualize the learned variance structure and demonstrate applications to image segmentation and denoising.

## 1  Introduction

Unsupervised learning has become an important tool for understanding biological information processing and building intelligent signal processing methods. Real biological systems however are much more robust and flexible than current artificial intelligence mostly due to a much more efficient representations used in biological systems. Therefore, unsupervised learning algorithms that capture more sophisticated representations can provide a better understanding of neural information processing and also provide improved algorithm for signal processing applications. For example, independent component analysis (ICA) can learn representations similar to simple cell receptive fields in visual cortex [1] and is also applied for feature extraction, image segmentation and denoising [2,3]. ICA can approximate statistics of natural image patches by Eq.(1,2), where X is the data and u is a source signal whose distribution is a product of sparse distributions like a generalized Laplacian distribution.

$$X = Au \qquad (1) \qquad\qquad P(u) = \prod P(u_i) \qquad (2)$$

But the representation learned by the ICA algorithm is relatively low-level. In biological systems there are more high-level representations such as contours, textures and objects, which are not well represented by the linear ICA model. ICA learns only linear dependency between pixels by finding strongly correlated linear

axis. Therefore, the modeling capability of ICA is quite limited. Previous approaches showed that one can learn more sophisticated high-level representations by capturing nonlinear dependencies in a post-processing step after the ICA step [4,5,6,7,8].

The focus of these efforts has centered on variance correlation in natural images. After ICA, a source signal is not linearly predictable from others. However, given variance dependencies, a source signal is still 'predictable' in a nonlinear manner. It is not possible to de-correlate this variance dependency using a linear transformation. Several researchers have proposed extensions to capture the nonlinear dependencies.

Portilla et al. used Gaussian Scale Mixture (GSM) to model variance dependency in wavelet domain. This model can learn variance correlation in source prior and showed improvement in image denoising [4]. But in this model, dependency is defined only between a subset of wavelet coefficients. Hyvarinen and Hoyer suggested using a special variance related distribution to model the variance correlated source prior. This model can learn grouping of dependent sources (Subspace ICA) or topographic arrangements of correlated sources (Topographic ICA) [5,6]. Similarly, Welling et al. suggested a product of expert model where each expert represents a variance correlated group [7]. The product form of the model enables applications to image denoising. But these models don't reveal higher-order structures explicitly.

Our model is motivated by Lewicki and Karklin who proposed a 2-stage model where the 1st stage is an ICA model (Eq. (3)) and the 2nd-stage is a linear generative model where another source v generates logarithmic variance for the 1st stage (Eq. (4)) [8]. This model captures variance dependency structure explicitly, but treating variance as an additional random variable introduces another level of complexity and requires several approximations. Thus, it is difficult to obtain a simple analytic PDF of source signal u and to apply the model for image processing problems.

$$P(u \mid \lambda) = c \exp\left(- \left| u / \lambda \right|^q \right) \quad (3) \qquad\qquad \log[\lambda] = Bv \quad (4)$$

We propose a hierarchical model based on ICA and a mixture of Laplacian distribution. Our model can be considered as a simplification of model in [8] by constraining v to be 0/1 random vector where only one element can be 1. Our model is computationally simpler but still can capture variance dependency. Experiments show that our model can reveal higher order structures similar to [8]. In addition, our model provides a simple parametric PDF of variance correlated priors, which is an important advantage for adaptive signal processing. Utilizing this, we demonstrate simple applications on image segmentation and image denoising. Our model provides an improved statistic model for natural images and can be used for other applications including feature extraction, image coding, or learning even higher order structures.

## 2   Modeling nonlinear dependencies

We propose a hierarchical or 2-stage model where the 1st stage is an ICA source signal model and the 2nd stage is modeled by a mixture model with different variances (figure 1). In natural images, the correlation of variance reflects different types of regularities in the real world. Such specialized regularities can be summarized as "context" information. To model the context dependent variance correlation, we use mixture models where Laplacian distributions with different variance represent different contexts. For each image patch, a context variable Z "selects" which Laplacian distribution will represent ICA source signal u. Laplacian distributions have 0-mean

but different variances. *The advantage of Laplacian distribution for modeling context is that we can model a sparse distribution using only one Laplacian distribution. But we need more than two Gaussian distributions to do the same thing.* Also conventional ICA is a special case of our model with one Laplacian. We define the mixture model and its learning algorithm in the next sections.

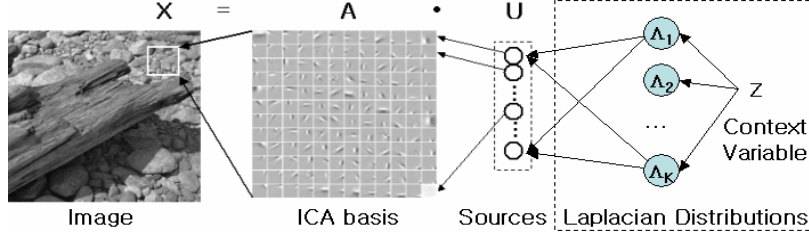

Figure 1: Proposed hierarchical model (1st stage is ICA generative model. 2nd stage is mixture of "context dependent" Laplacian distributions which model U. Z is a random variable that selects a Laplacian distribution that generates the given image patch)

## 2.1 Mixture of Laplacian Distribution

We define a PDF for mixture of M-dimensional Laplacian Distribution as Eq.(5), where N is the number of data samples, and K is the number of mixtures.

$$P(U \mid \Lambda, \Pi) = \prod_n^N P(\vec{u}_n \mid \Lambda, \Pi) = \prod_n^N \sum_k^K \pi_k P(\vec{u}_n \mid \vec{\lambda}_k) = \prod_n^N \sum_k^K \pi_k \prod_m^M \frac{1}{(2\lambda_{k,m})} \exp\left(-\frac{|u_{n,m}|}{\lambda_{k,m}}\right) \tag{5}$$

$\vec{u}_n = (u_{n,1}, u_{n,2}, \ldots, u_{n,M})$ : n-th data sample, $U = (\vec{u}_1, \vec{u}_2, \ldots, \vec{u}_i, \ldots, \vec{u}_N)$

$\vec{\lambda}_k = (\lambda_{k,1}, \lambda_{k,2}, \ldots, \lambda_{k,M})$ : Variance of k-th Laplacian distribution, $\Lambda = (\vec{\lambda}_1, \vec{\lambda}_2, \ldots, \vec{\lambda}_k, \ldots, \vec{\lambda}_K)$

$\pi_k$ : probability of Laplacian distribution k, $\Pi = (\pi_1, \ldots, \pi_K)$ and $\sum_k \pi_k = 1$

It is not easy to maximize Eq.(5) directly, and we use EM (expectation maximization) algorithm for parameter estimation. Here we introduce a new hidden context variable Z that represents which Laplacian k, is responsible for a given data point. Assuming we know the hidden variable Z, we can write the likelihood of data and Z as Eq.(6),

$$P(U, Z \mid \Lambda, \Pi) = \prod_n^N P(\vec{u}_n, Z \mid \Lambda, \Pi) = \prod_n^N \left[ \prod_k^K \left[ (\pi_k)^{z_k^n} \prod_m \left( \left(\frac{1}{2\lambda_{k,m}}\right)^{z_k^n} \cdot \exp\left(-z_k^n \frac{|u_{n,m}|}{\lambda_{k,m}}\right) \right) \right] \right] \tag{6}$$

$z_k^n$ : Hidden binary random variable, 1 if n-th data sample is generated from k-th Laplacian, 0 other wise. ($Z = (z_k^n)$ and $\sum_k z_k^n = 1$ for all n = 1…N)

## 2.2 EM algorithm for learning the mixture model

The EM algorithm maximizes the log likelihood of data averaged over hidden variable Z. The log likelihood and its expectation can be computed as in Eq.(7,8).

$$\log P(U,Z\mid\Lambda,\Pi)=\sum_{n,k}\left[z_k^n\log(\pi_k)+\sum_m z_k^n\left(\log(\frac{1}{2\lambda_{k,m}})-\left|\frac{u_{n,m}}{\lambda_{k,m}}\right|\right)\right] \tag{7}$$

$$E\{\log P(U,Z\mid\Lambda,\Pi)\}=\sum_{n,k}E\{z_k^n\}\left[\log(\pi_k)+\sum_m\left(\log(\frac{1}{2\lambda_{k,m}})-\left|\frac{u_{n,m}}{\lambda_{k,m}}\right|\right)\right] \tag{8}$$

The expectation in Eq.(8) can be evaluated, if we are given the data U and estimated parameters $\Lambda$ and $\Pi$. For $\Lambda$ and $\Pi$, EM algorithm uses current estimation $\Lambda'$ and $\Pi'$.

$$E\{z_k^n\}\equiv E\{z_k^n\mid U,\Lambda',\Pi'\}=\sum_{z_k^n=0}^{1}z_k^n P(z_k^n\mid u_n,\Lambda',\Pi')=P(z_k^n=1\mid u_n,\Lambda',\Pi')$$

$$=\frac{P(u_n\mid z_k^n=1,\Lambda',\Pi')P(z_k^n=1\mid\Lambda',\Pi')}{P(u_n\mid\Lambda',\Pi')} \tag{9}$$

$$=\frac{1}{P(u_n\mid\Lambda',\Pi')}\prod_m^M\frac{1}{2\lambda_{k,m}'}\exp(-\left|\frac{u_{n,m}}{\lambda_{k,m}'}\right|)\cdot\pi_k'=\frac{1}{c_n}\prod_m^M\frac{\pi_k'}{2\lambda_{k,m}'}\exp(-\left|\frac{u_{n,m}}{\lambda_{k,m}'}\right|)$$

Where the normalization constant can be computed as

$$c_n=P(u_n\mid\Lambda',\Pi')=\sum_k^K P(u_n\mid z_k^n,\Lambda',\Pi')P(z_k^n\mid\Lambda',\Pi')=\sum_{k=1}^{K}\pi_k\prod_{m=1}^{M}\frac{1}{(2\lambda_{k,m})}\exp(-\left|\frac{u_{n,m}}{\lambda_{k,m}}\right|) \tag{10}$$

The EM algorithm works by maximizing Eq.(8), given the expectation computed from Eq.(9,10). Eq.(9,10) can be computed using $\Lambda'$ and $\Pi'$ estimated in the previous iteration of EM algorithm. This is E-step of EM algorithm. Then in M-step of EM algorithm, we need to maximize Eq.(8) over parameter $\Lambda$ and $\Pi$.

First, we can maximize Eq.(8) with respect to $\Lambda$, by setting the derivative as 0.

$$\frac{\partial E\{\log P(U,Z\mid\Lambda,\Pi)\}}{\partial\lambda_{k,m}}=\sum_n E\{z_k^n\}\left[\left(-\frac{1}{\lambda_{k,m}}+\frac{|u_{n,m}|}{(\lambda_{k,m})^2}\right)\right]=0\quad\Rightarrow\quad\lambda_{k,m}=\frac{\sum_n E\{z_k^n\}\cdot|u_{n,m}|}{\sum_n E\{z_k^n\}} \tag{11}$$

Second, for maximization of Eq.(8) with respect to $\Pi$, we can rewrite Eq.(8) as below.

$$E\{\log P(U,Z\mid\Lambda,\Pi)\}=C+\sum_{n,k'}E\{z_{k'}^n\}\log(\pi_{k'}) \tag{12}$$

As we see, the derivative of Eq.(12) with respect to $\Pi$ cannot be 0. Instead, we need to use Lagrange multiplier method for maximization. A Lagrange function can be defined as Eq.(14) where $\rho$ is a Lagrange multiplier.

$$L(\Pi,\rho)=-\sum_{n,k'}E\{z_{k'}^n\}\log(\pi_{k'})+\rho(\sum_{k'}\pi_{k'}-1) \tag{13}$$

By setting the derivative of Eq.(13) to be 0 with respect to $\rho$ and $\Pi$, we can simply get the maximization solution with respect to $\Pi$. We just show the solution in Eq.(14).

$$\frac{\partial L(\Pi,\rho)}{\partial\rho}=0,\frac{\partial L(\Pi,\rho)}{\partial\Pi}=0\quad\Rightarrow\quad\pi_k=\left(\sum_n E\{z_k^n\}\right)/\left(\sum_k\sum_n E\{z_k^n\}\right) \tag{14}$$

Then the EM algorithm can be summarized as figure 2. For the convergence criteria, we can use the expectation of log likelihood, which can be calculated from Eq. (8).

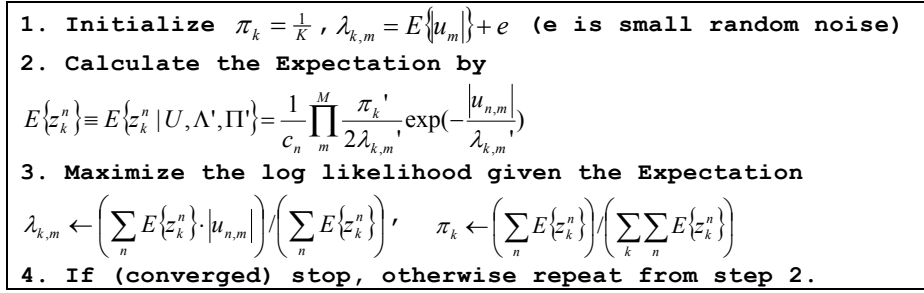

$$E\{z_k^n\} \equiv E\{z_k^n \mid U, \Lambda', \Pi'\} = \frac{1}{c_n} \prod_m^M \frac{\pi_k'}{2\lambda_{k,m}'} \exp(-\frac{|u_{n,m}|}{\lambda_{k,m}'})$$

**3. Maximize the log likelihood given the Expectation**

$$\lambda_{k,m} \leftarrow \left(\sum_n E\{z_k^n\} \cdot |u_{n,m}|\right) \bigg/ \left(\sum_n E\{z_k^n\}\right), \quad \pi_k \leftarrow \left(\sum_n E\{z_k^n\}\right) \bigg/ \left(\sum_k \sum_n E\{z_k^n\}\right)$$

**4. If (converged) stop, otherwise repeat from step 2.**

Figure 2: Outline of EM algorithm for Learning the Mixture Model

## 3 Experimental Results

Here we provide examples of image data and show how the learning procedure is performed for the mixture model. We also provide visualization of learned variances that reveal the structure of variance correlation and an application to image denoising.

### 3.1 Learning Nonlinear Dependencies in Natural images

As shown in figure 1, the 1st stage of the proposed model is simply the linear ICA. The ICA matrix A and W($=A^{-1}$) are learned by the FastICA algorithm [9]. We sampled $10^5$($=N$) data from 16x16 patches (256 dim.) of natural images and use them for both first and second stage learning. ICA input dimension is 256, and source dimension is set to be 160($=M$). The learned ICA basis is partially shown in figure 1. The 2nd stage mixture model is learned given the ICA source signals. In the 2nd stage the number of mixtures is set to 16, 64, or 256($=K$). Training by the EM algorithm is fast and several hundred iterations are sufficient for convergence (0.5 hour on a 1.7GHz Pentium PC).

For the visualization of learned variance, we adapted the visualization method from [8]. Each dimension of ICA source signal corresponds to an ICA basis (columns of A) and each ICA basis is localized in both image and frequency space. Then for each Laplacian distribution, we can display its variance vector as a set of points in image and frequency space. Each point can be color coded by variance value as figure 3.

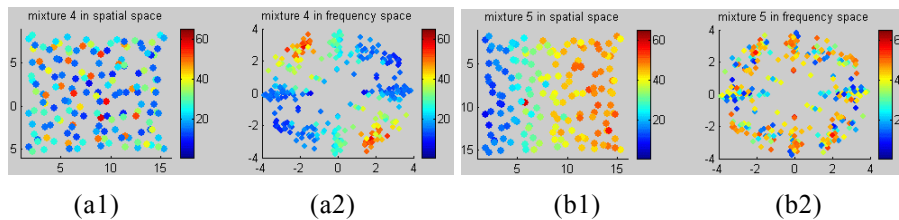

(a1)　　　　　(a2)　　　　　(b1)　　　　　(b2)

Figure 3: Visualization of learned variances (a1 and a2 visualize variance of Laplacian #4 and b1 and 2 show that of Laplacian #5. High variance value is mapped to red color and low variance is mapped to blue. In Laplacian #4, variances for diagonally oriented edges are high. But in Laplacian #5, variances for edges at spatially right position are high. Variance structures are related to *"contexts"* in the image. For example, Laplacian #4 explains image patches that have oriented textures or edges. Laplacian #5 captures patches where left side of the patch is clean but right side is filled with randomly oriented edges.)

A key idea of our model is that *we can mix up independent distributions to get non-linearly dependent distribution.* This modeling power can be shown by figure 4.

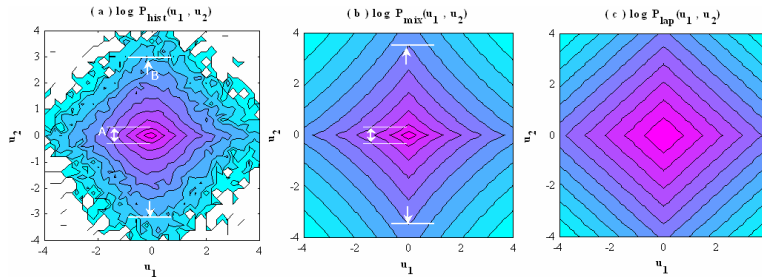

Figure 4: Joint distribution of nonlinearly dependent sources. ((a) is a joint histogram of 2 ICA sources, (b) is computed from learned mixture model, and (c) is from learned Laplacian model. In (a), *variance of $u_2$ is smaller than $u_1$ at center area (arrow A), but almost equal to $u_1$ at outside (arrow B)*. So the variance of $u_2$ is dependent on $u_1$. This nonlinear dependency is closely approximated by mixture model in (b), but not in (c).)

### 3.2 Unsupervised Image Segmentation

The idea behind our model is that the image can be modeled as mixture of different variance correlated "contexts". We show how the learned model can be used to classify different context by an unsupervised image segmentation task. Given learned model and data, we can compute the expectation of a hidden variable Z from Eq. (9). Then for an image patch, we can select a Laplacian distribution with highest probability, which is the most explaining Laplacian or "context". For segmentation, we use the model with 16 Laplacians. This enables abstract partitioning of images and we can visualize organization of images more clearly (figure 5).

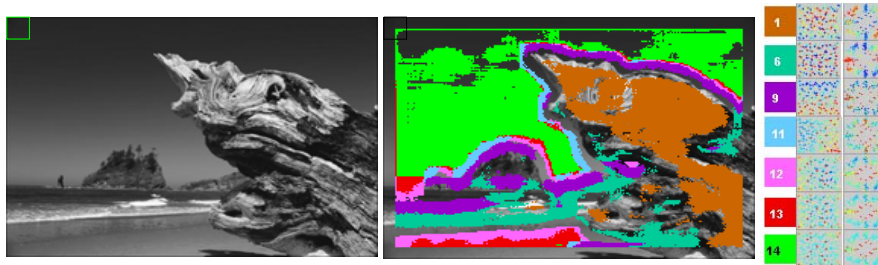

Figure 5: Unsupervised image segmentation (left is original image, middle is color labeled image, right image shows color coded Laplacians with variance structure. Each color corresponds to a Laplacian distribution, which represents surface or textural organization of underlying contexts. Laplacian #14 captures smooth surface and Laplacian #9 captures contrast between clear sky and textured ground scenes.)

### 3.3 Application to Image Restoration

The proposed mixture model provides a better parametric model of the ICA source distribution and hence an improved model of the image structure. An advantage is in the MAP (maximum a posterior) estimation of a noisy image. If we assume Gaussian noise n, the image generation model can be written as Eq.(15). Then, we can compute MAP estimation of ICA source signal u by Eq.(16) and reconstruct the original image.

$$X = Au + n \tag{15}$$

$$\hat{u} = \underset{u}{argmax} \log P(u \mid X, A) = \underset{u}{argmax} \big( \log P(X \mid u, A) + \log P(u) \big) \tag{16}$$

Since we assumed Gaussian noise, P(X|u,A) in Eq. (16) is Gaussian. P(u) in Eq. (16) can be modeled as a Laplacian or a mixture of Laplacian distribution. The mixture distribution can be approximated by a maximum explaining Laplacian. We evaluated 3 different methods for image restoration including ICA MAP estimation with simple Laplacian prior, same with Laplacian mixture prior, and the Wiener filter. Figure 6 shows an example and figure 7 summarizes the results obtained with different noise levels. As shown MAP estimation with the mixture prior performs better than the others in terms of SNR and SSIM (Structural Similarity Measure) [10].

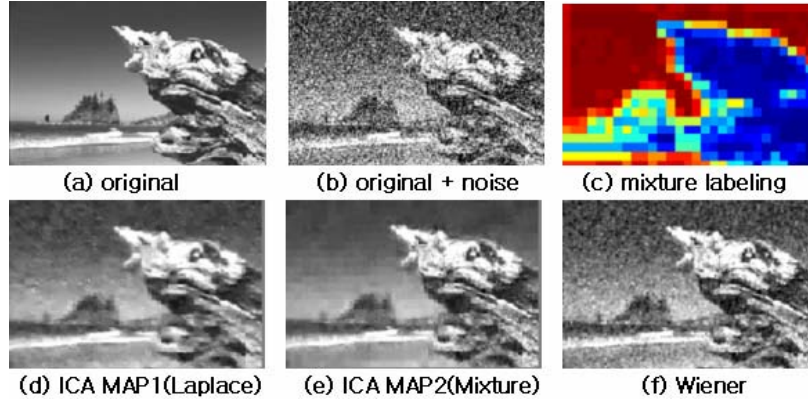

(a) original         (b) original + noise         (c) mixture labeling

(d) ICA MAP1(Laplace)      (e) ICA MAP2(Mixture)      (f) Wiener

Figure 6: Image restoration results (signal variance 1.0, noise variance 0.81)

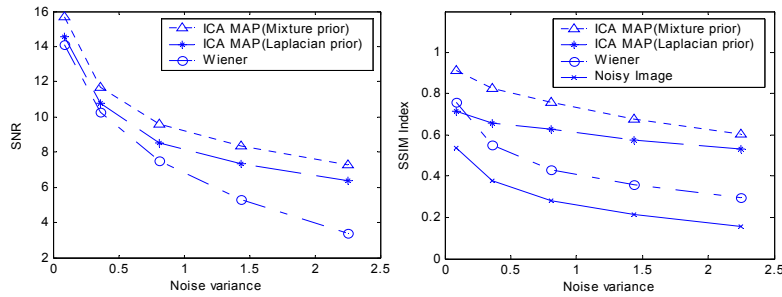

Figure 7: SNR and SSIM for 3 different algorithms (signal variance = 1.0)

## 4 Discussion

We proposed a mixture model to learn nonlinear dependencies of ICA source signals for natural images. The proposed mixture of Laplacian distribution model is a generalization of the conventional independent source priors and can model variance dependency given natural image signals. Experiments show that the proposed model can learn the variance correlated signals grouped as different mixtures and learn high-level structures, which are highly correlated with the underlying physical properties

captured in the image. Our model provides an analytic prior of nearly independent and variance-correlated signals, which was not viable in previous models [4,5,6,7,8].

The learned variances of the mixture model show structured localization in image and frequency space, which are similar to the result in [8]. Since the model is given no information about the spatial location or frequency of the source signals, we can assume that the dependency captured by the mixture model reveals regularity in the natural images. As shown in image labeling experiments, such regularities correspond to specific surface types (textures) or boundaries between surfaces.

The learned mixture model can be used to discover hidden contexts that generated such regularity or correlated signal groups. Experiments also show that the labeling of image patches is highly correlated with the object surface types shown in the image. The segmentation results show regularity across image space and strong correlation with high-level concepts.

Finally, we showed applications of the model for image restoration. We compare the performance with the conventional ICA MAP estimation and Wiener filter. Our results suggest that the proposed model outperforms other traditional methods. It is due to the estimation of the correlated variance structure, which provides an improved prior that has not been considered in other methods.

In our future work, we plan to exploit the regularity of the image segmentation result to lean more high-level structures by building additional hierarchies on the current model. Furthermore, the application to image coding seems promising.

## References

[1] A. J. Bell and T. J. Sejnowski, The 'Independent Components' of Natural Scenes are Edge Filters, *Vision Research*, 37(23):3327–3338, 1997.

[2] A. Hyvarinen, Sparse Code Shrinkage: Denoising of Nongaussian Data by Maximum Likelihood Estimation,Neural Computation, 11(7):1739-1768, 1999.

[3] T. Lee, M. Lewicki, and T. Sejnowski., ICA Mixture Models for unsupervised Classification of non-gaussian classes and automatic context switching in blind separation. PAMI, 22(10), October 2000.

[4] J. Portilla, V. Strela, M. J. Wainwright and E. P Simoncelli, Image Denoising using Scale Mixtures of Gaussians in the Wavelet Domain, IEEE Trans. On Image Processing, Vol.12, No. 11, 1338-1351, 2003.

[5] A. Hyvarinen, P. O. Hoyer. Emergence of phase and shift invariant features by decomposition of natural images into independent feature subspaces. Neurocomputing, 1999.

[6] A. Hyvarinen, P.O. Hoyer, Topographic Independent component analysis as a model of V1 Receptive Fields, Neurocomputing, Vol. 38-40, June 2001.

[7] M. Welling and G. E. Hinton, S. Osindero, Learning Sparse Topographic Representations with Products of Student-t Distributions, NIPS, 2002.

[8] M. S. Lewicki and Y. Karklin, Learning higher-order structures in natural images, Network: Comput. Neural Syst. 14 (August 2003) 483-499.

[9] A.Hyvarinen, P.O. Hoyer, Fast ICA matlab code.,
http://www.cis.hut.fi/projects/compneuro/extensions.html/

[10] Z. Wang, A. C. Bovik, H. R. Sheikh and E. P. Simoncelli, The SSIM Index for Image Quality Assessment, IEEE Transactions on Image Processing, vol. 13, no. 4, Apr. 2004.
